# Effects of Spatial and Temporal Contiguity on the Acquisition of Spatial Information

**Thea B. Ghiselli-Crippa and Paul W. Munro**
Department of Information Science and Telecommunications
University of Pittsburgh
Pittsburgh, PA 15260
*tbgst@sis.pitt.edu, munro@sis.pitt.edu*

## Abstract

Spatial information comes in two forms: direct spatial information (for example, retinal position) and indirect temporal contiguity information, since objects encountered sequentially are in general spatially close. The acquisition of spatial information by a neural network is investigated here. Given a spatial layout of several objects, networks are trained on a prediction task. Networks using temporal sequences with no direct spatial information are found to develop internal representations that show distances correlated with distances in the external layout. The influence of spatial information is analyzed by providing direct spatial information to the system during training that is either consistent with the layout or inconsistent with it. This approach allows examination of the relative contributions of spatial and temporal contiguity.

## 1 Introduction

Spatial information is acquired by a process of exploration that is fundamentally temporal, whether it be on a small scale, such as scanning a picture, or on a larger one, such as physically navigating through a building, a neighborhood, or a city. Continuous scanning of an environment causes locations that are spatially close to have a tendency to occur in temporal proximity to one another. Thus, a temporal associative mechanism (such as a Hebb rule) can be used in conjunction with continuous exploration to capture the spatial structure of the environment [1]. However, the actual process of building a cognitive map need not rely solely on temporal associations, since some spatial information is encoded in the sensory array (position on the retina and proprioceptive feedback). Laboratory studies show different types of interaction between the relative contributions of temporal and spatial contiguities to the formation of an internal representation of space. While Clayton and Habibi's [2] series of recognition priming experiments indicates that priming is controlled only by temporal associations, in the work of McNamara et al. [3] priming in recognition is observed only when space and time are both contiguous. In addition, Curiel and Radvansky's [4] work shows that the effects of spatial and temporal contiguity depend on whether location or identity information is emphasized during learning. Moreover, other experiments ([3]) also show how the effects clearly depend on the task and can be quite different if an explicitly spatial task is used (e.g., additive effects in location judgments).

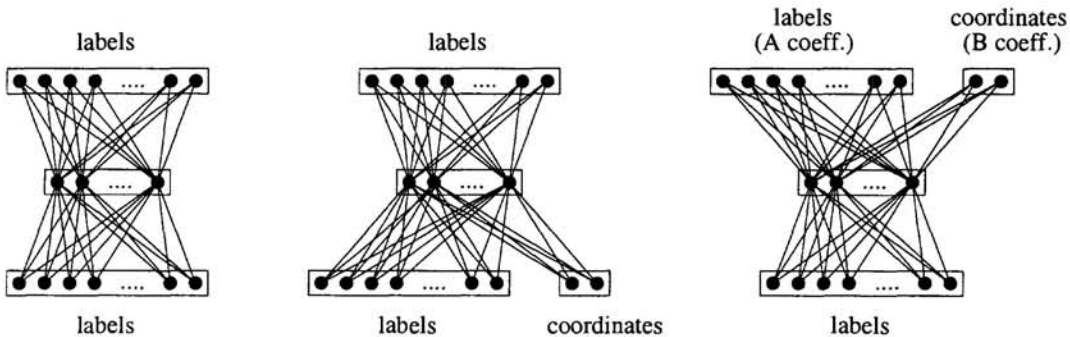

Figure 1: Network architectures: temporal-only network (left); spatio-temporal network with spatial units part of the input representation (center); spatio-temporal network with spatial units part of the output representation (right).

## 2    Network architectures

The goal of the work presented in this paper is to study the structure of the internal representations that emerge from the integration of temporal and spatial associations. An encoder-like network architecture is used (see Figure 1), with a set of $N$ input units and a set of $N$ output units representing $N$ nodes on a 2-dimensional graph. A set of $H$ units is used for the hidden layer. To include space in the learning process, additional spatial units are included in the network architecture. These units provide a representation of the spatial information directly available during the learning/scanning process. In the simulations described in this paper, two units are used and are chosen to represent the $(x, y)$ coordinates of the nodes in the graph. The spatial units can be included as part of the input representation or as part of the output representation (see Figure 1, center and right panels): both choices are used in the experiments, to investigate whether the spatial information could better benefit training as an input or as an output [5]. In the second case, the relative contribution of the spatial information can be directly manipulated by introducing weighting factors in the cost function being minimized. A two-term cost function is used, with a cross-entropy term for the $N$ label units and a squared error term for the 2 coordinate units,

$$E = A \left[ -\sum_{i=1}^{N} t_i log_2(r_i) + (1 - t_i)log_2(1 - r_i) \right] + B \left[ \frac{1}{2} \sum_{i=1}^{2} (t_i - r_i)^2 \right] \quad (1)$$

$r_i$ indicates the actual output of unit $i$ and $t_i$ its desired output. The relative influence of the spatial information is controlled by the coefficients $A$ and $B$.

## 3    Learning tasks

The left panel of Figure 2 shows an example of the type of layout used; the effective layout used in the study consists of $N = 28$ nodes. For each node, a set of neighboring nodes is defined, chosen on the basis of how an observer might scan the layout to learn the node labels and their (spatial) relationships; in Figure 2, the neighborhood relationships are represented by lines connecting neighboring nodes. From any node in the layout, the only allowed transitions are those to a neighbor, thus defining the set of node pairs used to train the network (66 pairs out of $C(28, 2) = 378$ possible pairs). In addition, the probability of occurrence of a particular transition is computed as a function of the distance to the corresponding neighbor. It is then possible to generate a sequence of visits to the network nodes, aimed at replicating the scanning process of a human observer studying the layout.

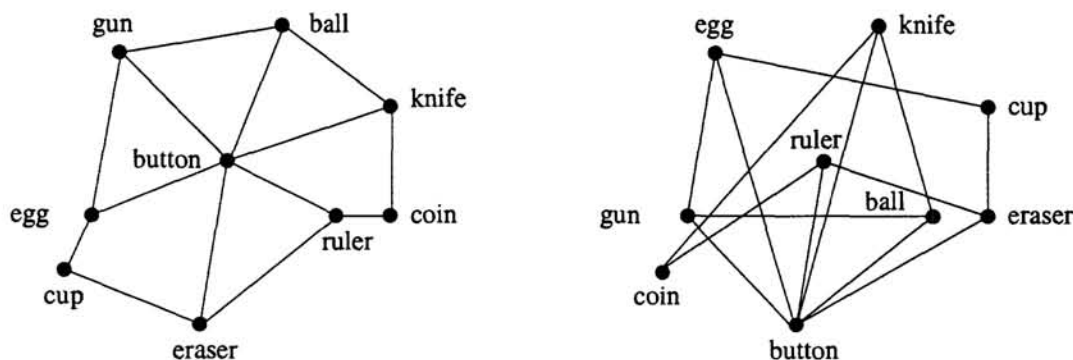

Figure 2: Example of a layout (left) and its permuted version (right). Links represent allowed transitions. A larger layout of 28 units was used in the simulations.

The basic learning task is similar to the grammar learning task of Servan-Schreiber et al. [6] and to the neighborhood mapping task described in [1] and is used to associate each of the $N$ nodes on the graph and its $(x, y)$ coordinates with the probability distribution of the transitions to its neighboring nodes. The mapping can be learned directly, by associating each node with the probability distribution of the transitions to all its neighbors: in this case, batch learning is used as the method of choice for learning the mapping. On the other hand, the mapping can be learned indirectly, by associating each node with itself and one of its neighbors, with online learning being the method of choice in this case; the neighbor chosen at each iteration is defined by the sequence of visits generated on the basis of the transition probabilities. Batch learning was chosen because it generally converges more smoothly and more quickly than online learning and gives qualitatively similar results. While the task and network architecture described in [1] allowed only for temporal association learning, in this study both temporal and spatial associations are learned simultaneously, thanks to the presence of the spatial units. However, the temporal-only (T-only) case, which has no spatial units, is included in the simulations performed for this study, to provide a benchmark for the evaluation of the results obtained with the spatio-temporal (S-T) networks.

The task described above allows the network to learn neighborhood relationships for which spatial and temporal associations provide consistent information, that is, nodes experienced contiguously in time (as defined by the sequence) are also contiguous in space (being spatial neighbors). To tease apart the relative contributions of space and time, the task is kept the same, but the data employed for training the network is modified: the same layout is used to generate the temporal sequence, but the $x, y$ coordinates of the nodes are randomly permuted (see right panel of Figure 2). If the permuted layout is then scanned following the same sequence of node visits used in the original version, the net effect is that the temporal associations remain the same, but the spatial associations change so that temporally neighboring nodes can now be spatially close or distant: the spatial associations are no longer consistent with the temporal associations. As Figure 4 illustrates, the training pairs (filled circles) all correspond to short distances in the original layout, but can have a distance anywhere in the allowable range in the permuted layout. Since the temporal and spatial distances were consistent in the original layout, the original spatial distance can be used as an indicator of temporal distance and Figure 4 can be interpreted as a plot of temporal distance vs. spatial distance for the permuted layout.

The simulations described in the following include three experimental conditions: temporal only (no direct spatial information available); space and time consistent (the spatial coordinates and the temporal sequence are from the same layout); space and time inconsistent (the spatial coordinates and the temporal sequence are from different layouts).

Hidden unit representations are compared using Euclidean distance (cosine and inner product measures give consistent results); the internal representation distances are also used to compute their correlation with Euclidean distances between nodes in the layout (original and permuted). The correlations increase with the number of hidden units for values of $H$ between 5 and 10 and then gradually taper off for values greater than 10. The results presented in the remainder of the paper all pertain to networks trained with $H = 20$ and with hidden units using a $tanh$ transfer function; all the results pertaining to S-T networks refer to networks with 2 spatial output units and cost function coefficients $A = 0.625$ and $B = 6.25$.

## 4   Results

Figure 3 provides a combined view of the results from all three experiments. The left panel illustrates the evolution of the correlation between internal representation distances and layout (original and permuted) distances. The right panel shows the distributions of the correlations at the end of training (1000 epochs). The first general result is that, when spatial information is available and consistent with the temporal information (original layout), the correlation between hidden unit distances and layout distances is consistently better than the correlation obtained in the case of temporal associations alone. The second general result is that, when spatial information is available but not consistent with the temporal information (permuted layout), the correlation between hidden unit distances and original layout distances (which represent temporal distances) is similar to that obtained in the case of temporal associations alone, except for the initial transient. When the correlation is computed with respect to the permuted layout distances, its value peaks early during training and then decreases rapidly, to reach an asymptotic value well below the other three cases. This behavior is illustrated in the boxplots in the right panel of Figure 3, which report the distribution of correlation values at the end of training.

### 4.1   Temporal-only vs. spatio-temporal

As a first step in this study, the effects of adding spatial information to the basic temporal associations used to train the network can be examined. Since the learning task is the same for both the T-only and the S-T networks except for the absence or presence of spatial information during training, the differences observed can be attributed to the additional spatial information available to the S-T networks. The higher correlation between internal representation distances and original layout distances obtained when spatial information is

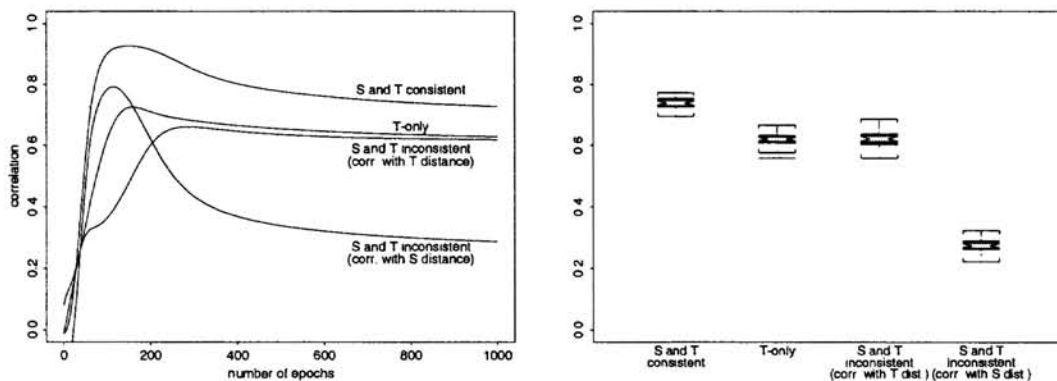

Figure 3: Evolution of correlation during training (0 - 1000 epochs) (left). Distributions of correlations at the end of training (1000 epochs) (right).

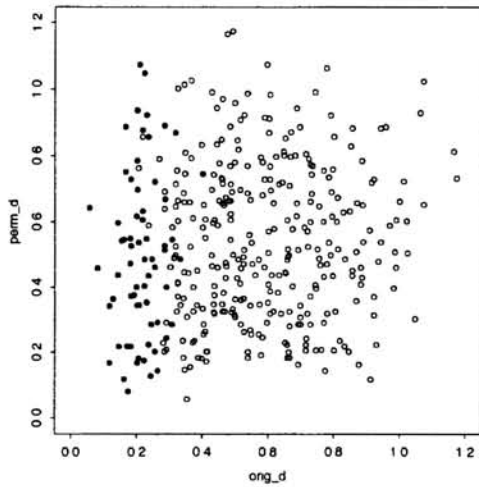

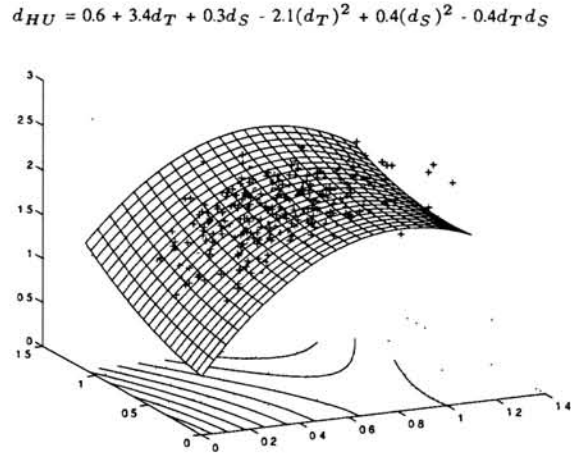

$$d_{HU} = 0.6 + 3.4d_T + 0.3d_S - 2.1(d_T)^2 + 0.4(d_S)^2 - 0.4d_T d_S$$

Figure 4: Distances in the original layout ($x$) vs. distances in the permuted layout ($y$). The 66 training pairs are identified by filled circles.

Figure 5: Similarities (Euclidean distances) between internal representations developed by a S-T network (after 300 epochs). Figure 4 projects the data points onto the $x, y$ plane.

available (see Figure 3) is apparent also when the evolution of the internal representations is examined. As Figure 6 illustrates, the presence of spatial information results in better generalization for the pattern pairs outside the training set. While the distances between training pairs are mapped to similar distances in hidden unit space for both the T-only and the S-T networks, the T-only network tends to cluster the non-training pairs into a narrow band of distances in hidden unit space. In the case of the S-T network instead, the hidden unit distances between non-training pairs are spread out over a wider range and tend to reflect the original layout distances.

## 4.2 Permuted layout

As described above, with the permuted layout it is possible to decouple the spatial and temporal contributions and therefore study the effects of each. A comprehensive view of the results at a particular point during training (300 epochs) is presented in Figure 5, where the $x, y$ plane represents temporal distance vs. spatial distance (see also Figure 4) and the $z$ axis represents the similarity between hidden unit representations. The figure also includes a quadratic regression surface fitted to the data points. The coefficients in the equation of the surface provide a quantitative measure of the relative contributions of spatial ($d_S$) and temporal distances ($d_T$) to the similarity between hidden unit representations ($d_{HU}$):

$$d_{HU} = k_0 + k_1 d_T + k_2 d_S + k_3 (d_T)^2 + k_4 (d_S)^2 + k_5 d_T d_S \tag{2}$$

In general, after the transient observed in early training (see Figure 3), the largest and most significant coefficients are found for $d_T$ and $(d_T)^2$, indicating a stronger dependence of $d_{HU}$ on temporal distance than on spatial distance.

The results illustrated in Figure 5 represent the situation at a particular point during training (300 epochs). Similar plots can be generated for different points during training, to study the evolution of the internal representations. A different view of the evolution process is provided by Figure 7, in which the data points are projected onto the $x, z$ plane (top panel) and the $y, z$ plane (bottom panel) at four different times during training. In the top panel,

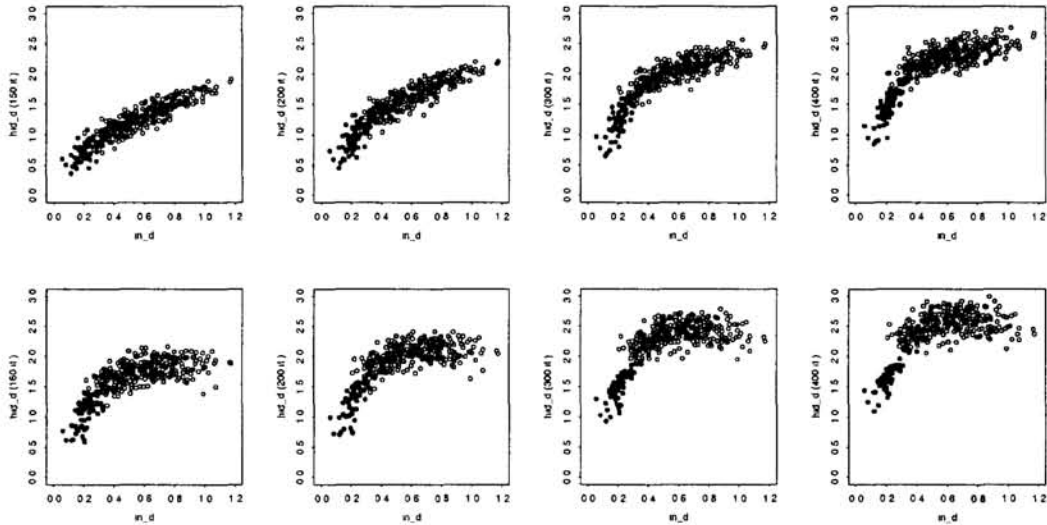

Figure 6: Internal representation distances vs. original layout distances: S-T network (top) vs. T-only network (bottom). The training pairs are identified by filled circles. The presence of spatial information results in better generalization for the pairs outside the training set.

the internal representation distances are plotted as a function of temporal distance (i.e., the spatial distance from the original layout), while in the bottom panel they are plotted as a function of spatial distance (from the permuted layout). The higher asymptotic correlation between internal representation distances and temporal distances, as opposed to spatial distances (see Figure 3), is apparent also from the examination of the evolutionary plots, which show an asymptotic behavior with respect to temporal distances (see Figure 7, top panel) very similar to the T-only case (see Figure 6, bottom panel).

## 5   Discussion

The first general conclusion that can be drawn from the examination of the results described in the previous section is that, when the spatial information is available and consistent with the temporal information (original layout), the similarity structure of the hidden unit representations is closer to the structure of the original layout than that obtained by using temporal associations alone. The second general conclusion is that, when the spatial information is available but not consistent with the temporal information (permuted layout), the similarity structure of the hidden unit representations seems to correspond to temporal more than spatial proximity. Figures 5 and 7 both indicate that temporal associations take precedence over spatial associations. This result is in agreement with the results described in [1], showing how temporal associations (plus some high-level constraints) significantly contribute to the internal representation of global spatial information. However, spatial information certainly is very beneficial to the (temporal) acquisition of a layout, as proven by the results obtained with the S-T network vs. the T-only network.

In terms of the model presented in this paper, the results illustrated in Figures 5 and 7 can be compared with the experimental data reported for recognition priming ([2], [3], [4]), with distance between internal representations corresponding to reaction time. The results of our model indicate that distances in both the spatially far and spatially close condition appear to be consistently shorter for the training pairs (temporally close) than for the non-training pairs (temporally distant), highlighting a strong temporal effect consistent with the data reported in [2] and [4] (for spatially far pairs) and in [3] (only for the spatially close

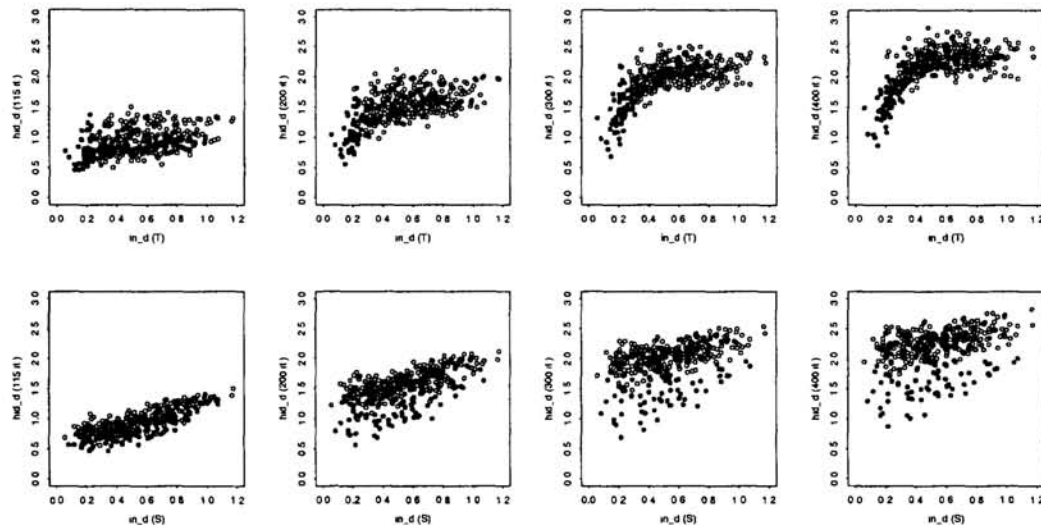

Figure 7: Internal representation distances vs. temporal distances (top) and vs. spatial distances (bottom) for a S-T network (permuted layout). The training pairs are identified by filled circles. The asymptotic behavior with respect to temporal distances (top panel) is similar to the T-only condition. The bottom panel indicates a weak dependence on spatial distances.

case). For the training pairs (temporally close), slightly shorter distances are obtained for spatially close pairs vs. spatially far pairs; this result does not provide support for the experimental data reported in either [3] (strong spatial effect) or [2] (no spatial effect). For the non-training pairs (temporally distant), long distances are found throughout, with no strong dependence on spatial distance; this effect is consistent with all the reported experimental data. Further simulations and statistical analyses are necessary for a more conclusive comparison with the experimental data.

## References

[1] Ghiselli-Crippa, T.B. & Munro, P.W. (1994). Emergence of global structure from local associations. In J.D. Cowan, G. Tesauro, & J. Alspector (Eds.), *Advances in Neural Information Processing Systems 6*, pp. 1101-1108. San Francisco, CA: Morgan Kaufmann.

[2] Clayton, K.N. & Habibi, A. (1991). The contribution of temporal contiguity to the spatial priming effect. *Journal of Experimental Psychology: Learning, Memory, and Cognition* 17:263-271.

[3] McNamara, T.P., Halpin, J.A. & Hardy, J.K. (1992). Spatial and temporal contributions to the structure of spatial memory. *Journal of Experimental Psychology: Learning, Memory, and Cognition* 18:555-564.

[4] Curiel, J.M. & Radvansky, G.A. (1998). Mental organization of maps. *Journal of Experimental Psychology: Learning, Memory, and Cognition* 24:202-214.

[5] Caruana, R. & de Sa, V.R. (1997). Promoting poor features to supervisors: Some inputs work better as outputs. In M.C. Mozer, M.I. Jordan, & T. Petsche (Eds.), *Advances in Neural Information Processing Systems 9*, pp. 389-395. Cambridge, MA: MIT Press.

[6] Servan-Schreiber, D., Cleeremans, A. & McClelland, J.L. (1989). Learning sequential structure in simple recurrent networks. In D.S. Touretzky (Ed.), *Advances in Neural Information Processing Systems 1*, pp. 643-652. San Mateo, CA: Morgan Kaufmann.
